# Random Projections for Manifold Learning

**Chinmay Hegde**
ECE Department
Rice University
ch3@rice.edu

**Michael B. Wakin**
EECS Department
University of Michigan
wakin@eecs.umich.edu

**Richard G. Baraniuk**
ECE Department
Rice University
richb@rice.edu

## Abstract

We propose a novel method for *linear* dimensionality reduction of manifold modeled data. First, we show that with a small number $M$ of *random projections* of sample points in $\mathbb{R}^N$ belonging to an unknown $K$-dimensional Euclidean manifold, the intrinsic dimension (ID) of the sample set can be estimated to high accuracy. Second, we rigorously prove that using only this set of random projections, we can estimate the structure of the underlying manifold. In both cases, the number of random projections required is linear in $K$ and logarithmic in $N$, meaning that $K < M \ll N$. To handle practical situations, we develop a greedy algorithm to estimate the smallest size of the projection space required to perform manifold learning. Our method is particularly relevant in distributed sensing systems and leads to significant potential savings in data acquisition, storage and transmission costs.

## 1 Introduction

Recently, we have witnessed a tremendous increase in the sizes of data sets generated and processed by acquisition and computing systems. As the volume of the data increases, memory and processing requirements need to correspondingly increase at the same rapid pace, and this is often prohibitively expensive. Consequently, there has been considerable interest in the task of effective modeling of high-dimensional observed data and information; such models must capture the structure of the information content in a concise manner.

A powerful data model for many applications is the geometric notion of a low-dimensional *manifold*. Data that possesses merely $K$ "intrinsic" degrees of freedom can be assumed to lie on a $K$-dimensional manifold in the high-dimensional ambient space. Once the manifold model is identified, any point on it can be represented using essentially $K$ pieces of information. Thus, algorithms in this vein of dimensionality reduction attempt to *learn* the structure of the manifold given high-dimensional training data.

While most conventional manifold learning algorithms are adaptive (i.e., data dependent) and non-linear (i.e., involve construction of a nonlinear mapping), a *linear, nonadaptive* manifold dimensionality reduction technique has recently been introduced that employs *random projections* [1]. Consider a $K$-dimensional manifold $\mathcal{M}$ in the ambient space $\mathbb{R}^N$ and its projection onto a random subspace of dimension $M = CK \log(N)$; note that $K < M \ll N$. The result of [1] is that the pairwise metric structure of sample points from $\mathcal{M}$ is preserved with high accuracy under projection from $\mathbb{R}^N$ to $\mathbb{R}^M$.

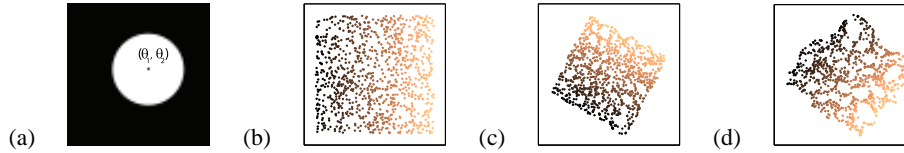

(a)          (b)          (c)          (d)

Figure 1: *Manifold learning using random projections. (a) Input data consisting of 1000 images of a shifted disk, each of size $N = 64 \times 64 = 4096$. (b) True $\theta_1$ and $\theta_2$ values of the sampled data. (c,d) Isomap embedding learned from (c) original data in $\mathbb{R}^N$, and (d) a randomly projected version of the data into $\mathbb{R}^M$ with $M = 15$.*

This result has far reaching implications. Prototypical devices that directly and inexpensively acquire random projections of certain types of data (signals, images, etc.) have been developed [2, 3]; these devices are hardware realizations of the mathematical tools developed in the emerging area of Compressed Sensing (CS) [4, 5]. The theory of [1] suggests that a wide variety of signal processing tasks can be performed *directly on the random projections* acquired by these devices, thus saving valuable sensing, storage and processing costs.

The advantages of random projections extend even to cases where the original data is available in the ambient space $\mathbb{R}^N$. For example, consider a wireless network of cameras observing a scene. To perform joint image analysis, the following steps might be executed:

1. **Collate**: Each camera node transmits its respective captured image (of size $N$) to a central processing unit.

2. **Preprocess**: The central processor estimates the *intrinsic dimension $K$* of the underlying image manifold.

3. **Learn**: The central processor performs a nonlinear embedding of the data points – for instance, using Isomap [6] – into a $K$-dimensional Euclidean space, using the estimate of $K$ from the previous step.

In situations where $N$ is large and communication bandwidth is limited, the dominating costs will be in the first transmission/collation step. On the one hand, to reduce the communication needs one may perform nonlinear image compression (such as JPEG) at each node before transmitting to the central processing. But this requires a good deal of processing power at each sensor, and the compression would have to be undone during the learning step, thus adding to overall computational costs. On the other hand, every camera could encode its image by computing (either directly or indirectly) a small number of random projections to communicate to the central processor. These random projections are obtained by linear operations on the data, and thus are cheaply computed. Clearly, in many situations it will be less expensive to store, transmit, and process such randomly projected versions of the sensed images. The question now becomes: how much information about the manifold is conveyed by these random projections, and is any advantage in analyzing such measurements from a manifold learning perspective?

In this paper, we provide theoretical and experimental evidence that reliable learning of a $K$-dimensional manifold can be performed not just in the high-dimensional ambient space $\mathbb{R}^N$ but also in an intermediate, much lower-dimensional random projection space $\mathbb{R}^M$, where $M = CK \log(N)$. See, for example, the toy example of Figure 1. Our contributions are as follows. First, we present a theoretical bound on the minimum number of measurements per sample point required to estimate the intrinsic dimension (ID) of the underlying manifold, up to an accuracy level comparable to that of the Grassberger-Procaccia algorithm [7, 8], a widely used geometric approach for dimensionality estimation. Second, we present a similar bound on the number of measurements $M$ required for Isomap [6] – a popular manifold learning algorithm – to be "reliably" used to discover the nonlinear structure of the manifold. In both cases, $M$ is shown to be linear in $K$ and logarithmic in $N$. Third, we formulate a procedure to determine, in practical settings, this minimum value of $M$ with no *a priori* information about the data points. This paves the way for a weakly adaptive, linear algorithm (**ML-RP**) for dimensionality reduction and manifold learning.

The rest of the paper is organized as follows. Section 2 recaps the manifold learning approaches we utilize. In Section 3 presents our main theoretical contributions, namely, the bounds on $M$ required to perform reliable dimensionality estimation and manifold learning from random projections. Sec-

tion 4 describes a new adaptive algorithm that estimates the minimum value of $M$ required to provide a faithful representation of the data so that manifold learning can be performed. Experimental results on a variety of real and simulated data are provided in Section 5. Section 6 concludes with discussion of potential applications and future work.

## 2   Background

An important input parameter for all manifold learning algorithms is the *intrinsic dimension* (ID) of a point cloud. We aim to embed the data points in as low-dimensional a space as possible in order to avoid the curse of dimensionality. However, if the embedding dimension is too small, then distinct data points might be collapsed onto the same embedded point. Hence a natural question to ask is: given a point cloud in $N$-dimensional Euclidean space, what is the dimension of the manifold that best captures the structure of this data set? This problem has received considerable attention in the literature and remains an active area of research [7, 9, 10].

For the purposes of this paper, we focus our attention on the Grassberger-Procaccia (GP) [7] algorithm for ID estimation. This is a widely used geometric technique that takes as input the set of pairwise distances between sample points. It then computes the *scale-dependent correlation dimension* of the data, defined as follows.

**Definition 2.1** *Suppose $X = (x_1, x_2, ..., x_n)$ is a finite dataset of underlying dimension $K$. Define*

$$C_n(r) = \frac{1}{n(n-1)} \sum_{i \neq j} \mathbf{I}_{\|x_i - x_j\| < r},$$

*where $\mathbf{I}$ is the indicator function. The scale-dependent correlation dimension of $X$ is defined as*

$$\widehat{D}_{\mathrm{corr}}(r_1, r_2) = \frac{\log C_n(r_1) - \log C_n(r_2)}{\log r_1 - \log r_2}.$$

The best possible approximation to $K$ (call this $\widehat{K}$) is obtained by fixing $r_1$ and $r_2$ to the biggest range over which the plot is linear and the calculating $D_{\mathrm{corr}}$ in that range. There are a number of practical issues involved with this approach; indeed, it has been shown that geometric ID estimation algorithms based on finite sampling yield biased estimates of intrinsic dimension [10, 11]. In our theoretical derivations, we do not attempt to take into account this bias; instead, we prove that the effect of running the GP algorithm on a sufficient number of random projections produces a dimension estimate that well-approximates the GP estimate obtained from analyzing the original point cloud.

The estimate $\widehat{K}$ of the ID of the point cloud is used by nonlinear manifold learning algorithms (e.g., Isomap [6], Locally Linear Embedding (LLE) [12], and Hessian Eigenmaps [13], among many others) to generate a $\widehat{K}$-dimensional coordinate representation of the input data points. Our main analysis will be centered around Isomap. Isomap attempts to preserve the *metric structure* of the manifold, i.e., the set of pairwise geodesic distances of any given point cloud sampled from the manifold. In essence, Isomap approximates the geodesic distances using a suitably defined graph and performs classical multidimensional scaling (MDS) to obtain a reduced $K$-dimensional representation of the data [6]. A key parameter in the Isomap algorithm is the *residual variance*, which is equivalent to the stress function encountered in classical MDS. The residual variance is a measure of how well the given dataset can be embedded into a Euclidean space of dimension $K$. In the next section, we prescribe a specific number of measurements per data point so that performing Isomap on the randomly projected data yields a residual variance that is arbitrarily close to the variance produced by Isomap on the original dataset.

We conclude this section by revisiting the results derived in [1], which form the basis for our development. Consider the effect of projecting a smooth $K$-dimensional manifold residing in $\mathbb{R}^N$ onto a random $M$-dimensional subspace (isomorphic to $\mathbb{R}^M$). If $M$ is sufficiently large, a stable near-isometric embedding of the manifold in the lower-dimensional subspace is ensured. The key advantage is that $M$ needs only to be *linear* in the intrinsic dimension of the manifold $K$. In addition, $M$ depends only logarithmically on other properties of the manifold, such as its volume, curvature, etc. The result can be summarized in the following theorem.

**Theorem 2.2** [1] *Let $\mathcal{M}$ be a compact $K$-dimensional manifold in $\mathbb{R}^N$ having volume $V$ and condition number $1/\tau$. Fix $0 < \epsilon < 1$ and $0 < \rho < 1$. Let $\Phi$ be a random orthoprojector[1] from $\mathbb{R}^N$ to $\mathbb{R}^M$ and*

$$M \geq O\left(\frac{K \log(NV\tau^{-1})\log(\rho^{-1})}{\epsilon^2}\right). \tag{1}$$

*Suppose $M < N$. Then, with probability exceeding $1 - \rho$, the following statement holds: For every pair of points $x, y \in \mathcal{M}$, and $i \in \{1, 2\}$,*

$$(1 - \epsilon)\sqrt{\frac{M}{N}} \leq \frac{d_i(\Phi x, \Phi y)}{d_i(x, y)} \leq (1 + \epsilon)\sqrt{\frac{M}{N}}. \tag{2}$$

*where $d_1(x, y)$ (respectively, $d_2(x, y)$) stands for the geodesic (respectively, $\ell_2$) distance between points $x$ and $y$.*

The condition number $\tau$ controls the local, as well as global, curvature of the manifold – the smaller the $\tau$, the less well-conditioned the manifold with higher "twistedness" [1]. Theorem 2.2 has been proved by first specifying a finite high-resolution sampling on the manifold, the nature of which depends on its intrinsic properties; for instance, a planar manifold can be sampled coarsely. Then the Johnson-Lindenstrauss Lemma [14] is applied to these points to guarantee the so-called "isometry constant" $\epsilon$, which is nothing but (2).

## 3 Bounds on the performance of ID estimation and manifold learning algorithms under random projection

We saw above that random projections essentially ensure that the metric structure of a high-dimensional input point cloud (i.e., the set of all pairwise distances between points belonging to the dataset) is preserved up to a distortion that depends on $\epsilon$. This immediately suggests that geometry-based ID estimation and manifold learning algorithms could be applied to the lower-dimensional, randomly projected version of the dataset.

The first of our main results establishes a sufficient dimension of random projection $M$ required to maintain the fidelity of the estimated correlation dimension using the GP algorithm. The proof of the following is detailed in [15].

**Theorem 3.1** *Let $\mathcal{M}$ be a compact $K$-dimensional manifold in $\mathbb{R}^N$ having volume $V$ and condition number $1/\tau$. Let $X = \{x_1, x_2, ...\}$ be a sequence of samples drawn from a* uniform *density supported on $\mathcal{M}$. Let $\widehat{K}$ be the dimension estimate of the GP algorithm on $X$ over the range $(r_{\min}, r_{\max})$. Let $\beta = \ln(r_{\max}/r_{\min})$. Fix $0 < \delta < 1$ and $0 < \rho < 1$. Suppose the following condition holds:*

$$r_{max} < \tau/2 \tag{3}$$

*Let $\Phi$ be a random orthoprojector from $\mathbb{R}^N$ to $\mathbb{R}^M$ with $M < N$ and*

$$M \geq O\left(\frac{K \log(NV\tau^{-1})\log(\rho^{-1})}{\beta^2\delta^2}\right). \tag{4}$$

*Let $\widehat{K}_\Phi$ be the estimated correlation dimension on $\Phi X$ in the projected space* over the range *$(r_{min}\sqrt{M/N}, r_{max}\sqrt{M/N})$. Then, $\widehat{K}_\Phi$ is bounded by:*

$$(1 - \delta)\widehat{K} \leq \widehat{K}_\Phi \leq (1 + \delta)\widehat{K} \tag{5}$$

*with probability exceeding $1 - \rho$.*

Theorem 3.1 is a worst-case bound and serves as a sufficient condition for stable ID estimation using random projections. Thus, if we choose a sufficiently small value for $\delta$ and $\rho$, we are guaranteed estimation accuracy levels as close as desired to those obtained with ID estimation in the original signal space. Note that the bound on $\widehat{K}_\Phi$ is *multiplicative*. This implies that in the worst case, the

number of projections required to estimate $\widehat{K}_\Phi$ very close to $\widehat{K}$ (say, within integer roundoff error) becomes higher with increasing manifold dimension $K$.

The second of our main results prescribes the minimum dimension of random projections required to maintain the residual variance produced by Isomap in the projected domain within an arbitrary *additive* constant of that produced by Isomap with the full data in the ambient space. This proof of this theorem [15] relies on the proof technique used in [16].

**Theorem 3.2** *Let $\mathcal{M}$ be a compact $K$-dimensional manifold in $\mathbb{R}^N$ having volume $V$ and condition number $1/\tau$. Let $X = \{x_1, x_2, ..., x_n\}$ be a finite set of samples drawn from a sufficiently fine density supported on $\mathcal{M}$. Let $\Phi$ be a random orthoprojector from $\mathbb{R}^N$ to $\mathbb{R}^M$ with $M < N$. Fix $0 < \epsilon < 1$ and $0 < \rho < 1$. Suppose*

$$M \geq O\left(\frac{K \log(NV\tau^{-1})\log(\rho^{-1})}{\epsilon^2}\right).$$

*Define the* diameter $\Gamma$ *of the dataset as follows:*

$$\Gamma = \max_{1 \leq i,j \leq n} d_{iso}(x_i, x_j)$$

*where $d_{iso}(x, y)$ stands for the Isomap estimate of the geodesic distance between points $x$ and $y$. Define $R$ and $R_\Phi$ to be the residual variances obtained when Isomap generates a $K$-dimensional embedding of the original dataset $X$ and projected dataset $\Phi X$ respectively. Under suitable constructions of the Isomap connectivity graphs, $R_\Phi$ is bounded by:*

$$R_\Phi < R + C\Gamma^2 \epsilon$$

*with probability exceeding $1 - \rho$. $C$ is a function only on the number of sample points $n$.*

Since the choice of $\epsilon$ is arbitrary, we can choose a large enough $M$ (which is still only logarithmic in $N$) such that the residual variance yielded by Isomap on the randomly projected version of the dataset is arbitrarily close to the variance produced with the data in the ambient space. Again, this result is derived from a worst-case analysis. Note that $\Gamma$ acts as a measure of the scale of the dataset. In practice, we may enforce the condition that the data is normalized (i.e., every pairwise distance calculated by Isomap is divided by $\Gamma$). This ensures that the $K$-dimensional embedded representation is contained within a ball of unit norm centered at the origin.

Thus, we have proved that with only an $M$-dimensional projection of the data (with $M \ll N$) we can perform ID estimation and subsequently learn the structure of a $K$-dimensional manifold, up to accuracy levels obtained by conventional methods. In Section 4, we utilize these sufficiency results to motivate an algorithm for performing practical manifold structure estimation using random projections.

## 4 How many random projections are enough?

In practice, it is hard to know or estimate the parameters $V$ and $\tau$ of the underlying manifold. Also, since we have no *a priori* information regarding the data, it is impossible to fix $\widehat{K}$ and $R$, the outputs of GP and Isomap on the point cloud in the ambient space. Thus, often, we may not be able fix a definitive value for $M$. To circumvent this problem we develop the following empirical procedure that we dub it **ML-RP** for *manifold learning using random projections*.

We initialize $M$ to a small number, and compute $M$ random projections of the data set $X = \{x_1, x_2, ..., x_n\}$ (here $n$ denotes the number of points in the point cloud). Using the set $\Phi X = \{\Phi x : x \in X\}$, we estimate the intrinsic dimension using the GP algorithm. This estimate, say $\widehat{K}$, is used by the Isomap algorithm to produce an embedding into $\widehat{K}$-dimensional space. The residual variance produced by this operation is recorded. We then increment $M$ by 1 and repeat the entire process. The algorithm terminates when the residual variance obtained is smaller than some tolerance parameter $\delta$. A full length description is provided in Algorithm 1.

The essence of ML-RP is as follows. A sufficient number $M$ of random projections is determined by a nonlinear procedure (i.e., sequential computation of Isomap residual variance) so that conventional

**Algorithm 1** ML-RP
___
   $M \leftarrow 1$
   $\Phi \leftarrow$ Random orthoprojector of size $M \times N$.
   **while** residual variance $\geq \delta$ **do**
      Run the GP algorithm on $\Phi X$.
      Use ID estimate $(\widehat{K})$ to perform Isomap on $\Phi X$.
      Calculate residual variance.
      $M \leftarrow M + 1$
      Add one row to $\Phi$
   **end while**
   **return** $M$
   **return** $\widehat{K}$
___

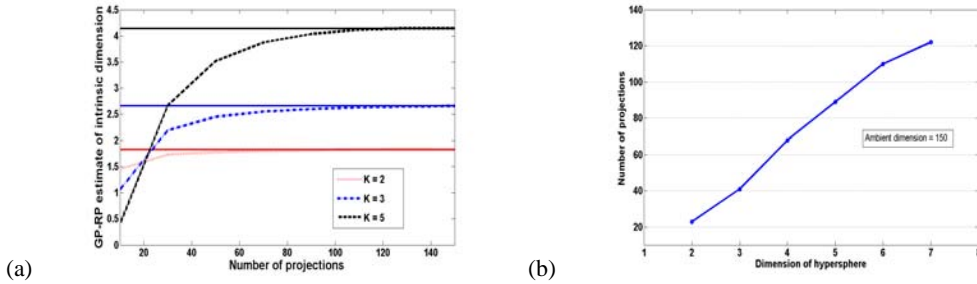

(a)                                             (b)

Figure 2: *Performance of ID estimation using GP as a function of random projections. Sample size n = 1000, ambient dimension N = 150. (a) Estimated intrinsic dimension for underlying hyperspherical manifolds of increasing dimension. The solid line indicates the value of the ID estimate obtained by GP performed on the original data. (b) Minimum number of projections required for GP to work with 90% accuracy as compared to GP on native data.*

manifold learning does almost as well on the projected dataset as the original. On the other hand, the random linear projections provide a faithful representation of the data in the geodesic sense. In this manner, ML-RP helps determine the number of rows that $\Phi$ requires in order to act as an operator that preserves metric structure. Therefore, ML-RP can be viewed as an adaptive method for linear reduction of data dimensionality. It is only weakly adaptive in the sense that only the stopping criterion for ML-RP is determined by monitoring the nature of the projected data.

The results derived in Section 3 can be viewed as convergence proofs for ML-RP. The existence of a certain minimum number of measurements for any chosen error value $\delta$ ensures that eventually, $M$ in the ML-RP algorithm is going to become high enough to ensure "good" Isomap performance. Also, due to the built-in parsimonious nature of ML-RP, we are ensured to not "overmeasure" the manifold, i.e., just the requisite numbers of projections of points are obtained.

## 5   Experimental results

This section details the results of simulations of ID estimation and subsequent manifold learning on real and synthetic datasets. First, we examine the performance of the GP algorithm on random projections of $K$-dimensional dimensional hyperspheres embedded in an ambient space of dimension $N = 150$. Figure 2(a) shows the variation of the dimension estimate produced by GP as a function of the number of projections $M$. The sampled dataset in each of the cases is obtained from drawing $n = 1000$ samples from a uniform distribution supported on a hypersphere of corresponding dimension. Figure 2(b) displays the minimum number of projections per sample point required to estimate the scale-dependent correlation dimension directly from the random projections, up to 10% error, when compared to GP estimation on the original data.

We observe that the ID estimate stabilizes quickly with increasing number of projections, and indeed converges to the estimate obtained by running the GP algorithm on the original data. Figure 2(b) illustrates the variation of the minimum required projection dimension $M$ vs. $K$, the intrinsic dimen-

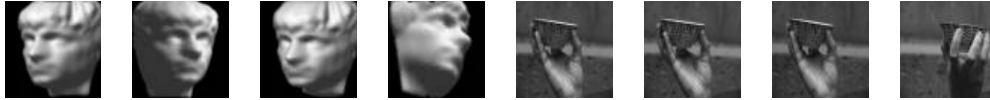

Figure 3: *Standard databases. Ambient dimension for the face database N = 4096; ambient dimension for the hand rotation databases N = 3840.*

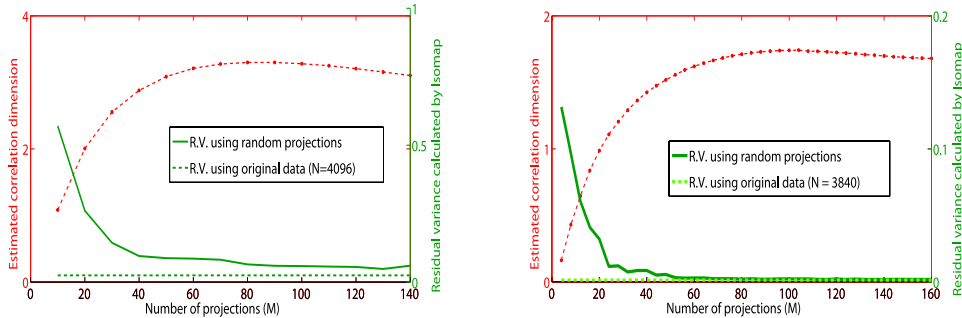

Figure 4: *Performance of ML-RP on the above databases. (left) ML-RP on the face database ($N = 4096$). Good approximates are obtained for $M > 50$. (right) ML-RP on the hand rotation database ($N = 3840$). For $M > 60$, the Isomap variance is indistinguishable from the variance obtained in the ambient space.*

sion of the underlying manifold. We plot the intrinsic dimension of the dataset against the minimum number of projections required such that $\widehat{K}_\Phi$ is within 10% of the conventional GP estimate $\widehat{K}$ (this is equivalent to choosing $\delta = 0.1$ in Theorem 3.1). We observe the predicted linearity (Theorem 3.1) in the variation of $M$ vs $K$.

Finally, we turn our attention to two common datasets (Figure 3) found in the literature on dimension estimation – the face database[2] [6], and the hand rotation database [17].[3] The face database is a collection of 698 artificial snapshots of a face ($N = 64 \times 64 = 4096$) varying under 3 degrees of freedom: 2 angles for pose and 1 for lighting dimension. The signals are therefore believed to reside on a 3D manifold in an ambient space of dimension 4096. The hand rotation database is a set of 90 images ($N = 64 \times 60 = 3840$) of rotations of a hand holding an object. Although the image appearance manifold is ostensibly one-dimensional, estimators in the literature always overestimate its ID [11].

Random projections of each sample in the databases were obtained by computing the inner product of the image samples with an increasing number of rows of the random orthoprojector $\Phi$. We note that in the case of the face database, for $M > 60$, the Isomap variance on the randomly projected points closely approximates the variance obtained with full image data. This behavior of convergence of the variance to the best possible value is even more sharply observed in the hand rotation database, in which the two variance curves are indistinguishable for $M > 60$. These results are particularly encouraging and demonstrate the validity of the claims made in Section 3.

## 6   Discussion

Our main theoretical contributions in this paper are the explicit values for the lower bounds on the minimum number of random projections required to perform ID estimation and subsequent manifold learning using Isomap, with high guaranteed accuracy levels. We also developed an empirical greedy algorithm (ML-RP) for practical situations. Experiments on simple cases, such as uniformly generated hyperspheres of varying dimension, and more complex situations, such as the image databases displayed in Figure 3, provide sufficient evidence of the nature of the bounds described above.

The method of random projections is thus a powerful tool for ensuring the stable embedding of low-dimensional manifolds into an intermediate space of reasonable size. The motivation for developing results and algorithms that involve random measurements of high-dimensional data is significant, particularly due to the increasing attention that Compressive Sensing (CS) has received recently. It is now possible to think of settings involving a huge number of low-power devices that inexpensively capture, store, and transmit a very small number of measurements of high-dimensional data. ML-RP is applicable in all such situations. In situations where the bottleneck lies in the transmission of the data to the central processing node, ML-RP provides a simple solution to the manifold learning problem and ensures that with minimum transmitted amount of information, effective manifold learning can be performed. The metric structure of the projected dataset upon termination of ML-RP closely resembles that of the original dataset with high probability; thus, ML-RP can be viewed as a novel adaptive algorithm for finding an efficient, reduced representation of data of very large dimension.

## Footnotes

[1]Such a matrix is formed by orthogonalizing $M$ vectors of length $N$ having, for example, i.i.d. Gaussian or Bernoulli distributed entries.

[2]*http://isomap.stanford.edu*

[3]*http://vasc.ri.cmu.edu//idb/html/motion/hand/index.html*.  Note that we use a subsampled version of the database used in the literature, both in terms of resolution of the image and sampling of the manifold.

# References

[1] R. G. Baraniuk and M. B. Wakin. Random projections of smooth manifolds. 2007. To appear in *Foundations of Computational Mathematics*.

[2] M. B. Wakin, J. N. Laska, M. F. Duarte, D. Baron, S. Sarvotham, D. Takhar, K. F. Kelly, and R. G. Baraniuk. An architecture for compressive imaging. In *IEEE International Conference on Image Processing (ICIP)*, pages 1273–1276, Oct. 2006.

[3] S. Kirolos, J.N. Laska, M.B. Wakin, M.F. Duarte, D.Baron, T. Ragheb, Y. Massoud, and R.G. Baraniuk. Analog-to-information conversion via random demodulation. In *Proc. IEEE Dallas Circuits and Systems Workshop (DCAS)*, 2006.

[4] E. J. Candès, J. Romberg, and T. Tao. Robust uncertainty principles: Exact signal reconstruction from highly incomplete frequency information. *IEEE Trans. Info. Theory*, 52(2):489–509, Feb. 2006.

[5] D. L. Donoho. Compressed sensing. *IEEE Trans. Info. Theory*, 52(4):1289–1306, September 2006.

[6] J. B. Tenenbaum, V.de Silva, and J. C. Landford. A global geometric framework for nonlinear dimensionality reduction. *Science*, 290:2319–2323, 2000.

[7] P. Grassberger and I. Procaccia. Measuring the strangeness of strange attractors. *Physica D Nonlinear Phenomena*, 9:189–208, 1983.

[8] J. Theiler. Statistical precision of dimension estimators. *Physical Review A*, 41(6):3038–3051, 1990.

[9] F. Camastra. Data dimensionality estimation methods: a survey. *Pattern Recognition*, 36:2945–2954, 2003.

[10] J. A. Costa and A. O. Hero. Geodesic entropic graphs for dimension and entropy estimation in manifold learning. *IEEE Trans. Signal Processing*, 52(8):2210–2221, August 2004.

[11] E. Levina and P. J. Bickel. Maximum likelihood estimation of intrinsic dimension. In *Advances in NIPS*, volume 17. MIT Press, 2005.

[12] S. Roweis and L. Saul. Nonlinear dimensionality reduction by locally linear embedding. *Science*, 290:2323–2326, 2000.

[13] D. Donoho and C. Grimes. Hessian eigenmaps: locally linear embedding techniques for high dimensional data. *Proc. of National Academy of Sciences*, 100(10):5591–5596, 2003.

[14] Sanjoy Dasgupta and Anupam Gupta. An elementary proof of the JL lemma. Technical Report TR-99-006, University of California, Berkeley, 1999.

[15] C. Hegde, M. B. Wakin, and R. G. Baraniuk. Random projections for manifold learning - proofs and analysis. Technical Report TREE 0710, Rice University, 2007.

[16] M. Bernstein, V. de Silva, J. Langford, and J. Tenenbaum. Graph approximations to geodesics on embedded manifolds, 2000. Technical report, Stanford University.

[17] B. Kégl. Intrinsic dimension estimation using packing numbers. In *Advances in NIPS*, volume 14. MIT Press, 2002.
